# Permitted and Forbidden Sets in Symmetric Threshold-Linear Networks

**Richard H.R. Hahnloser and H. Sebastian Seung**
Dept. of Brain & Cog. Sci., MIT
Cambridge, MA 02139 USA
rh@ai.mit.edu, seung@mit.edu

## Abstract

Ascribing computational principles to neural feedback circuits is an important problem in theoretical neuroscience. We study symmetric threshold-linear networks and derive stability results that go beyond the insights that can be gained from Lyapunov theory or energy functions. By applying linear analysis to subnetworks composed of coactive neurons, we determine the stability of potential steady states. We find that stability depends on two types of eigenmodes. One type determines global stability and the other type determines whether or not multistability is possible. We can prove the equivalence of our stability criteria with criteria taken from quadratic programming. Also, we show that there are permitted sets of neurons that can be coactive at a steady state and forbidden sets that cannot. Permitted sets are clustered in the sense that subsets of permitted sets are permitted and supersets of forbidden sets are forbidden. By viewing permitted sets as memories stored in the synaptic connections, we can provide a formulation of long-term memory that is more general than the traditional perspective of fixed point attractor networks.

A Lyapunov-function can be used to prove that a given set of differential equations is convergent. For example, if a neural network possesses a Lyapunov-function, then for almost any initial condition, the outputs of the neurons converge to a stable steady state. In the past, this stability-property was used to construct attractor networks that associatively recall memorized patterns. Lyapunov theory applies mainly to symmetric networks in which neurons have monotonic activation functions [1, 2]. Here we show that the restriction of activation functions to threshold-linear ones is not a mere limitation, but can yield new insights into the computational behavior of recurrent networks (for completeness, see also [3]).

We present three main theorems about the neural responses to constant inputs. The first theorem provides necessary and sufficient conditions on the synaptic weight matrix for the existence of a globally asymptotically stable set of fixed points. These conditions can be expressed in terms of *copositivity*, a concept from quadratic programming and linear complementarity theory. Alternatively, they can be expressed in terms of certain eigenvalues and eigenvectors of submatrices of the synaptic weight matrix, making a connection to linear systems theory. The theorem guarantees that

the network will produce a steady state response to any constant input. We regard this response as the computational output of the network, and its characterization is the topic of the second and third theorems.

In the second theorem, we introduce the idea of *permitted* and *forbidden* sets. Under certain conditions on the synaptic weight matrix, we show that there exist sets of neurons that are "forbidden" by the recurrent synaptic connections from being coactivated at a stable steady state, no matter what input is applied. Other sets are "permitted," in the sense that they can be coactivated for some input. The same conditions on the synaptic weight matrix also lead to conditional multistability, meaning that there exists an input for which there is more than one stable steady state. In other words, forbidden sets and conditional multistability are inseparable concepts.

The existence of permitted and forbidden sets suggests a new way of thinking about memory in neural networks. When an input is applied, the network must select a set of active neurons, and this selection is constrained to be one of the permitted sets. Therefore the permitted sets can be regarded as memories stored in the synaptic connections.

Our third theorem states that there are constraints on the groups of permitted and forbidden sets that can be stored by a network. No matter which learning algorithm is used to store memories, active neurons cannot arbitrarily be divided into permitted and forbidden sets, because subsets of permitted sets have to be permitted and supersets of forbidden sets have to be forbidden.

# 1 Basic definitions

Our theory is applicable to the network dynamics

$$\frac{dx_i}{dt} + x_i = \left[ b_i + \sum_j W_{ij} x_j \right]^+ \tag{1}$$

where $[u]^+ = \max\{u, 0\}$ is a rectification nonlinearity and the synaptic weight matrix is symmetric, $W_{ij} = W_{ji}$. The dynamics can also be written in a more compact matrix-vector form as $\dot{x} + x = [b + Wx]^+$. The state of the network is $x$. An input to the network is an arbitrary vector $b$. An output of the network is a steady state $\underline{x}$ in response to $b$. The existence of outputs and their relationship to the input are determined by the synaptic weight matrix $W$.

A vector $v$ is said to be nonnegative, $v \geq 0$, if all of its components are nonnegative. The nonnegative orthant $\{v : v \geq 0\}$ is the set of all nonnegative vectors. It can be shown that any trajectory starting in the nonnegative orthant remains in the nonnegative orthant. Therefore, for simplicity we will consider initial conditions that are confined to the nonnegative orthant $x \geq 0$.

# 2 Global asymptotic stability

**Definition 1** A steady state $\underline{x}$ is *stable* if for all initial conditions sufficiently close to $\underline{x}$, the state trajectory remains close to $\underline{x}$ for all later times.

A steady state is *asymptotically stable* if for all initial conditions sufficiently close to $\underline{x}$, the state trajectory converges to $\underline{x}$.

A set of steady states is *globally asymptotically stable* if from almost all initial

conditions, state trajectories converge to one of the steady states. Exceptions are of measure zero.

**Definition 2** A *principal submatrix* $A$ of a square matrix $B$ is a square matrix that is constructed by deleting a certain set of rows and the corresponding columns of $B$.

The following theorem establishes necessary and sufficient conditions on $W$ for global asymptotic stability.

**Theorem 1** *If $W$ is symmetric, then the following conditions are equivalent:*

1. *All nonnegative eigenvectors of all principal submatrices of $I - W$ have positive eigenvalues.*

2. *The matrix $I-W$ is copositive. That is, $x^T(I-W)x > 0$ for all nonnegative $x$, except $x = 0$.*

3. *For all b, the network has a nonempty set of steady states that are globally asymptotically stable.*

**Proof sketch:**

- $(1) \Rightarrow (2)$. Let $v^*$ be the minimum of $v^T(I - W)v$ over nonnegative $v$ on the unit sphere. If (2) is false, the minimum value is less than or equal to zero. It follows from Lagrange multiplier methods that the nonzero elements of $v^*$ comprise a nonnegative eigenvector of the corresponding principal submatrix of $W$ with eigenvalue greater than or equal to unity.

- $(2) \Rightarrow (3)$. By the copositivity of $I-W$, the function $L = \frac{1}{2}x^T(I-W)x - b^Tx$ is lower bounded and radially unbounded. It is also nonincreasing under the network dynamics in the nonnegative orthant, and constant only at steady states. By the Lyapunov stability theorem, the stable steady states are globally asymptotically stable. In the language of optimization theory, the network dynamics converges to a local minimum of $L$ subject to the nonnegativity constraint $x \geq 0$.

- $(3) \Rightarrow (1)$. Suppose that (1) is false. Then there exists a nonnegative eigenvector of a principal submatrix of $W$ with eigenvalue greater than or equal to unity. This can be used to construct an unbounded trajectory of the dynamics.■

The meaning of these stability conditions is best appreciated by comparing with the analogous conditions for the purely linear network obtained by dropping the rectification from (1). In a linear network, all eigenvalues of $W$ would have to be smaller than unity to ensure asymptotic stability. Here only nonnegative eigenvectors are able to grow without bound, due to the rectification, so that only their eigenvalues must be less than unity. All principal submatrices of $W$ must be considered, because different sets of feedback connections are active, depending on the set of neurons that are above threshold. In a linear network, $I - W$ would have to be positive definite to ensure asymptotic stability, but because of the rectification, here this condition is replaced by the weaker condition of copositivity.

The conditions of Theorem 1 for global asymptotic stability depend only on $W$, but not on $b$. On the other hand, steady states do depend on $b$. The next lemma says that the mapping from input to output is surjective.

**Lemma 1** *For any nonnegative vector $v \geq 0$ there exists an input $b$, such that $v$ is a steady state of equation 1 with input $b$.*

**Proof:** Define $c = v - \Sigma W \Sigma v$, where $\Sigma = diag(\sigma_1, ..., \sigma_N)$ and $\sigma_i = 1$ if $v_i > 0$ and $\sigma_i = 0$ if $v_i = 0$. Choose $b_i = c_i$ for $v_i > 0$ and $b_i = -1 - (\Sigma W \Sigma v)_i$ for $v_i = 0$. ∎

This Lemma states that any nonnegative vector can be realized as a fixed point. Sometimes this fixed point is stable, such as in networks subject to Theorem 1 in which only a single neuron is active. Indeed, the principal submatrix of $I - W$ corresponding to a single active neuron corresponds to a diagonal elements, which according to (1) must be positive. Hence it is always possible to activate only a single neuron at an asymptotically stable fixed point. However, as will become clear from the following Theorem, not all nonnegative vectors can be realized as an asymptotically stable fixed point.

## 3 Forbidden and permitted sets

The following characterizations of stable steady states are based on the interlacing Theorem [4]. This Theorem says that if $A$ is a $n - 1$ by $n - 1$ principal submatrix of a $n$ by $n$ symmetric matrix $B$, then the eigenvalues of $A$ fall in between the eigenvalues of $B$. In particular, the largest eigenvalue of $A$ is always smaller than the largest eigenvalue of $B$.

**Definition 3** *A set of neurons is* permitted *if the neurons can be coactivated at an asymptotically stable steady state for some input $b$. On the other hand, a set of neurons is* forbidden, *if they cannot be coactivated at an asymptotically stable steady state no matter what the input $b$.*

Alternatively, we might have defined a permitted set as a set for which the corresponding square sub-matrix of $I - W$ has only positive eigenvalues. And, similarly, a forbidden set could be defined as a set for which there is at least one non-positive eigenvalue. It follows from Theorem 1 that if the matrix $I - W$ is copositive, then the eigenvectors corresponding to non-positive eigenvalues of forbidden sets have to have both positive and non-positive components.

**Theorem 2** *If the matrix $I - W$ is copositive, then the following statements are equivalent:*

1. *The matrix $I - W$ is not positive definite.*

2. *There exists a forbidden set.*

3. *The network is conditionally multistable. That is, there exists an input $b$ such that there is more than one stable steady state.*

**Proof sketch:**

- $(1) \Rightarrow (2)$. $I - W$ is not positive definite and so there can be no asymptotically stable steady state in which all neurons are active, e.g. the set of all neurons is forbidden.

- $(2) \Rightarrow (3)$. Denote the forbidden set with $k$ active neurons by $\Sigma$. Without loss of generality, assume that the principal submatrix of $I - W$ corresponding to $\Sigma$ has $k - 1$ positive eigenvalues and only one non-positive eigenvalue (by virtue of the interlacing theorem and the fact that the diagonal elements of $I - W$ must be positive, there is always a subset of $\Sigma$, for which

this is true). By choosing $b_i > 0$ for neurons $i$ belonging to $\Sigma$ and $b_j \ll 0$ for neurons $j$ not belonging to $\Sigma$, the quadratic Lyapunov function $L$ defined in Theorem 1 forms a saddle in the nonnegative quadrant defined by $\Sigma$. The saddle point is the point where $L$ restricted to the hyperplane defined by the $k-1$ positive eigenvalues reaches its minimum. But because neurons can be initialized to lower values of $L$ on either side of the hyperplane and because $L$ is non-increasing along trajectories, there is no way trajectories can cross the hyperplane. In conclusion, we have constructed an input $b$ for which the network is multistable.

- $(3) \Rightarrow (1)$. Suppose that (1) is false. Then for all $b$ the Lyapunov function $L$ is convex and so has only a single local minimum in the convex domain $x \geq 0$. This local minimum is also the global minimum. The dynamics must converge to this minimum.∎

If $I - W$ is positive definite, then a symmetric threshold-linear network has a unique steady state. This has been shown previously [5]. The next Theorem is an expansion of this result, stating an equivalent condition using the concept of permitted sets.

**Theorem 3** *If $W$ is symmetric, then the following conditions are equivalent:*

1. *The matrix $I - W$ is positive definite.*

2. *All sets are permitted.*

3. *For all $b$ there is a unique steady state, and it is stable.*

**Proof:**

- $(1) \Rightarrow (2)$. If $I - W$ is positive definite, then it is copositive. Hence (1) in Theorem 2 is false and so (2) in Theorem 2 is false, e.g. all set are permitted.
- $(2) \Rightarrow (1)$. Suppose (1) is false, so the set of all neurons active must be forbidden, not all sets are permitted.
- $(1) \Longleftrightarrow (3)$. See [5].∎

The following Theorem characterizes the forbidden and the permitted sets.

**Theorem 4** *Any subset of a permitted set is permitted. Any superset of a forbidden set is forbidden.*

**Proof:** According to the interlacing Theorem, if the smallest eigenvalue of a symmetric matrix is positive, then so are the smallest eigenvalues of all its principal submatrices. And, if the smallest eigenvalue of a principal submatrix is negative, then so is the smallest eigenvalue of the original matrix.∎

## 4 An example - the ring network

A symmetric threshold-linear network with local excitation and larger range inhibition has been studied in the past as a model for how simple cells in primary visual cortex obtain their orientation tuning to visual stimulation [6, 7]. Inspired by these results, we have recently built an electronic circuit containing a ring network, using analog VLSI technology [3]. We have argued that the fixed tuning width of the neurons in the network arises because active sets consisting of more than a fixed

number of contiguous neurons are forbidden. Here we give a more detailed account of this fact and provide a surprising result about the existence of some spurious permitted sets.

Let the synaptic matrix of a 10 neuron ring-network be translationally invariant. The connection between neurons $i$ and $j$ is given by $W_{ij} = -\beta + \alpha_0 \delta_{ij} + \alpha_1 (\delta_{i,j+1} + \delta_{i+1,j}) + \alpha_2 (\delta_{i,j+2} + \delta_{i+2,j})$, where $\beta$ quantifies global inhibition, $\alpha_0$ self-excitation, $\alpha_1$ first-neighbor lateral excitation and $\alpha_2$ second-neighbor lateral excitation. In Figure 1 we have numerically computed the permitted sets of this network, with the parameters taken from [3], e.g. $\alpha_0 = 0$ $\alpha_1 = 1.1$ $\alpha_2 = 1$ $\beta = 0.55$. The permitted sets were determined by diagonalising the $2^{10}$ square sub-matrices of $I - W$ and by classifying the eigenvalues corresponding to nonnegative eigenvectors. The Figure 1 shows the resulting parent permitted sets (those that have no permitted supersets). Consistent with the finding that such ring-networks can explain contrast invariant tuning of V1 cells and multiplicative response modulation of parietal cells, we found that there are no permitted sets that consist of more than 5 contiguous active neurons. However, as can be seen, there are many non-contiguous permitted sets that could in principle be activated by exciting neurons in white and strongly inhibiting neurons in black.

Because the activation of the spurious permitted sets requires highly specific input (inhibition of high spatial frequency), it can be argued that the presence of the spurious permitted sets is not relevant for the normal operation of the ring network, where inputs are typically tuned and excitatory (such as inputs from LGN to primary visual cortex).

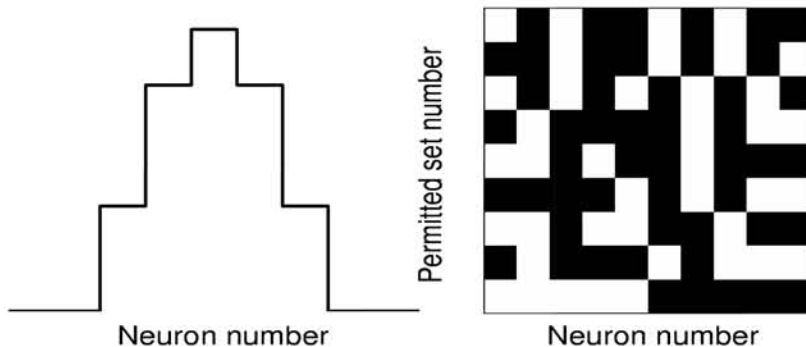

Figure 1: Left: Output of a ring network of 10 neurons to uniform input (random initial condition). Right: The 9 parent permitted sets (x-axis: neuron number, y-axis: set number). White means that a neurons belongs to a set and black means that it does not. Left-right and translation symmetric parent permitted sets of the ones shown have been excluded. The first parent permitted set (first row from the bottom) corresponds to the output on the left.

# 5  Discussion

We have shown that pattern memorization in threshold linear networks can be viewed in terms of permitted sets of neurons, e.g. sets of neurons that can be coactive at a steady state. According to this definition, the memories are stored by the synaptic weights, independently of the inputs. Hence, this concept of memory does not suffer from input-dependence, as would be the case for a definition of

memory based on the fixed points of the dynamics.

Pattern retrieval is strongly constrained by the input. A typical input will not allow for the retrieval of arbitrary stored permitted sets. This comes from the fact that multistability is not just dependent on the existence of forbidden sets, but also on the input (theorem 2). For example, in the ring network, positive input will always retrieve permitted sets consisting of a group of contiguous neurons, but not any of the spurious permitted sets, Figure 1. Generally, multistability in the ring network is only possible when more than a single neuron is excited.

Notice that threshold-linear networks can behave as traditional attractor networks when the inputs are represented as initial conditions of the dynamics. For example, by fixing $b = 1$ and initializing a copositive network with some input, the permitted sets unequivocally determine the stable fixed points. Thus, in this case, the notion of permitted sets is no different from fixed point attractors. However, the hierarchical grouping of permitted sets (Theorem 4) becomes irrelevant, since there can be only one attractive fixed point per hierarchical group defined by a parent permitted set.

The fact that no permitted set can have a forbidden subset represents a constraint on the possible computations of symmetric networks. However, this constraint does not have to be viewed as an undesired limitation. On the contrary, being aware of this constraint may lead to a deeper understanding of learning algorithms and representations for constraint satisfaction problems. We are reminded of the history of perceptrons, where the insight that they can only solve linearly separable classification problems led to the invention of multilayer perceptrons and backpropagation. In a similar way, grouping problems that do not obey the natural hierarchy inherent in symmetric networks, might necessitate the introduction of hidden neurons to realize the right geometry. For the interested reader, see also [8] for a simple procedure of how to store a given family of possibly overlapping patterns as permitted sets.

# References

[1] J. J. Hopfield. Neurons with graded response have collective properties like those of two-state neurons. *Proc. Natl. Acad. Sci. USA*, 81:3088–3092, 1984.

[2] M.A. Cohen and S. Grossberg. Absolute stability of global pattern formation and parallel memory storage by competitive neural networks. *IEEE Transactions on Systems, Man and Cybernetics*, 13:288–307, 1983.

[3] Richard H.R. Hahnloser, Rahul Sarpeshkar, Misha Mahowald, Rodney J. Douglas, and Sebastian Seung. Digital selection and ananlog amplification coexist in a silicon circuit inspired by cortex. *Nature*, 405:947–51, 2000.

[4] R.A. Horn and C.R. Johnson. *Matrix analysis*. Cambridge University Press, 1985.

[5] J. Feng and K.P. Hadeler. Qualitative behaviour of some simple networks. *J. Phys. A:*, 29:5019–5033, 1996.

[6] R. Ben-Yishai, R. Lev Bar-Or, and H. Sompolinsky. Theory of orientation tuning in visual cortex. *Proc. Natl. Acad. Sci. USA*, 92:3844–3848, 1995.

[7] R.J. Douglas, C. Koch, M.A. Mahowald, K.A.C. Martin, and H. Suarez. Recurrent excitation in neocortical circuits. *Science*, 269:981–985, 1995.

[8] Xie Xiaohui, Richard H.R. Hahnloser, and Sebastian Seung. Learning winner-take-all competition between groups of neurons in lateral inhibitory networks. In *Proceedings of NIPS2001 - Neural Information Processing Systems: Natural and Synthetic*, 2001.
